# Coastal Navigation with Mobile Robots

**Nicholas Roy and Sebastian Thrun**
School of Computer Science
Carnegie Mellon University
Pittsburgh, PA 15213
{*nicholas.roy|sebastian.thrun*}@*cs.cmu.edu*

## Abstract

The problem that we address in this paper is how a mobile robot can plan in order to arrive at its goal with minimum uncertainty. Traditional motion planning algorithms often assume that a mobile robot can track its position reliably, however, in real world situations, reliable localization may not always be feasible. Partially Observable Markov Decision Processes (POMDPs) provide one way to maximize the certainty of reaching the goal state, but at the cost of computational intractability for large state spaces.

The method we propose explicitly models the uncertainty of the robot's position as a state variable, and generates trajectories through the augmented pose-uncertainty space. By minimizing the positional uncertainty at the goal, the robot reduces the likelihood it becomes lost. We demonstrate experimentally that coastal navigation reduces the uncertainty at the goal, especially with degraded localization.

## 1 Introduction

For an operational mobile robot, it is essential to prevent becoming lost. Early motion planners assumed that a robot would never be lost – that a robot could always know its position via dead reckoning without error [7]. This assumption proved to be untenable due to the small and inevitable inconsistencies in actual robot motion; robots that rely solely on dead reckoning for their position estimates lose their position quickly. Mobile robots now perform position tracking using a combination of sensor data and odometry [2, 10, 5].

However, the robot's ability to track its position can vary considerably with the robot's position in the environment. Some parts of the environment may lack good features for localization [11]. Other parts of the environment can have a large number of dynamic features (for example, people) that can mislead the localization system. Motion planners rarely, if ever, take the robot's position tracking *ability* into consideration. As the robot's localization suffers, the likelihood that the robot becomes lost increases, and as a consequence, the robot is less likely to complete the given trajectory.

Most localization systems therefore compensate by adding environment-specific knowledge to the localization system, or by adding additional sensing capabilities to the robot, to guarantee that the robot can complete every possible path. In general, however, such alterations to the position tracking abilities of the robot have limitations, and an alternative scheme must be used to ensure that the robot can navigate with maximum reliability. The conventional planners represent one end of a spectrum of approaches (figure 1), in that a plan can be computed easily, but at the cost of not modelling localization performance.

At opposite end of the spectrum is the Partially Observable Markov Decision Process

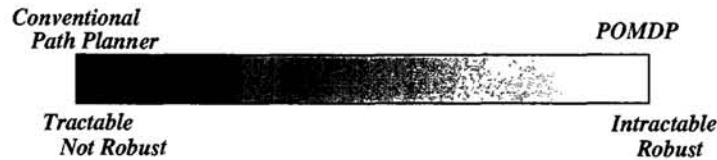

**Figure 1**: The continuum of possible approaches to the motion planning, from the robust but intractable POMDP, to the potentially failure-prone but real-time conventional planners. Coastal navigation lies in the middle of this spectrum.

(POMDP). POMDPs in a sense are the brass ring of planning with uncertainty; a POMDP policy will make exactly the right kind of compromise between conventional optimality considerations and certainty of achieving the goal state. Many people have examined the use of POMDPs for mobile robot navigation [5, 6, 8]. However, computing a POMDP solution is computationally intractable (PSPACE-hard) for large state systems – a mobile robot operating in the real world often has millions of possible states. As a result, many of the mobile robot POMDP solutions have made simplifying assumptions about the world in order to reduce the state space size. Many of these assumptions do not scale to larger environments or robots. In contrast, our hypothesis is that only a small number of the dimensions of the uncertainty matter, and that we can augment the state with these dimensions to approximate a solution to the POMDP.

The coastal navigation model developed in this paper represents a tradeoff between robust trajectories and computational tractability, and is inspired by traditional navigation of ships. Ships often use the coasts of continents for navigation in the absence of better tools such as GPS, since being close to the land allows sailors to determine with high accuracy where they are. The success of this method results from coast lines containing enough information in their structure for accurate localization. By navigating sufficiently close to areas of the map that have high information content, the likelihood of getting lost can be minimized.

## 2  Modelling Uncertainty

The problem that we address in this paper is how a mobile robot can plan in order to arrive at its goal with minimum uncertainty. Throughout this discussion, we will be assuming a known map of the environment [9]. The position, $\mathbf{x}$, of the robot is given as the location $(x, y)$ and direction $\theta$, defined over a space $\mathbf{X} = (X, Y, \Theta)$. Our localization method is a grid-based implementation of Markov localization [3, 5]. This method represents the robot's belief in its current position using a 3-dimensional grid over $\mathbf{X} = (X, Y, \Theta)$, which allows for a discrete approximation of arbitrary probability distributions. The probability that the robot has a particular pose $\mathbf{x}$ is given by the probability $p(\mathbf{x})$.

**State Augmentation**   We can extend the state of the robot from the 3-dimensional pose space to an augmented pose-uncertainty space. We can represent the uncertainty of the robot's positional distribution as the entropy,

$$H(P_{\mathbf{X}}) = - \int_{\mathbf{X}} p(\mathbf{x}) \log(p(\mathbf{x})) \, d\mathbf{x} \tag{1}$$

We therefore represent the state space of the robot as the tuple

$$\begin{aligned} \mathbf{S} &= \langle x, y, \theta, H(x, y, \theta) \rangle \\ &= \langle \mathbf{x}, H(\mathbf{x}) \rangle \end{aligned}$$

**State Transitions**   In order to construct a plan between two points in the environment, we need to be able to represent the effect of the robot's sensing and moving actions. The implementation of Markov localization provides the following equations for the tracking

the robot's pose from $\mathbf{x}$ to $\mathbf{x}'$:

$$p(\mathbf{x}'|u) = \int_{\mathbf{x}} p(\mathbf{x}'|\mathbf{x}, u)p(\mathbf{x})d\mathbf{x} \tag{2}$$

$$p(\mathbf{x}'|\mathbf{z}) = \alpha p(\mathbf{z}|\mathbf{x})p(\mathbf{x}) \tag{3}$$

These equations are taken from [3, 12], where equation (2) gives the prediction phase of localization (after motion $u$), and equation (3) gives the update phase of localization (after receiving observation $\mathbf{z}$). $\alpha$ is a normalizing constant. We extend these equations to the fourth dimension as follows:

$$p(\mathbf{s}|u) = \langle p(\mathbf{x}|u), H(p(\mathbf{x}|u)) \rangle \tag{4}$$

$$p(\mathbf{s}|\mathbf{z}) = \langle p(\mathbf{x}|\mathbf{z}), H(p(\mathbf{x}|\mathbf{z})) \rangle \tag{5}$$

## 3  Planning

Equations (4) and (5) provide a mechanism for tracking the robot's state, and in fact contain redundant information, since the extra state variable $H(\mathbf{x})$ is also contained in the probability distribution $p(\mathbf{x})$. However, in order to make the planning problem tractable, we cannot in fact maintain the probabilistic sensing model. To do so would put the planning problem firmly in the domain of POMDPs, with the associated computational intractability. Instead, we make a simplifying assumption, that is, that the positional probability distribution of the robot can be represented at all times by a Gaussian centered at the mean $\mathbf{x}$. This allows us to approximate the positional distribution with a single statistic, the entropy. In POMDP terms, we using the assumption of Gaussian distributions to compress the belief space to a single dimension. We can now represent the positional probability distribution completely with the vector $\mathbf{s}$, since the width of the Gaussian is represented by the entropy $H(\mathbf{x})$.

More importantly, the simplifying assumption allows us to track the state of the robot deterministically. Although the state transitions are stochastic (as in equation (4)), the observations are not. At any point in time, the sensors identify the true state of the system, with some certainty given by $H(p(\mathbf{x}|\mathbf{z}))$. This allows us to compress the state transitions into a single rule:

$$p(\mathbf{s}|u) = \langle p(\mathbf{x}|u), H(p(\mathbf{x}|u, \mathbf{z})) \rangle \tag{6}$$

The final position of the robot depends only on the motion command $u$ and can be identified by sensing $\mathbf{z}$. However, the uncertainty of the pose, $H(p(\mathbf{x}|u, \mathbf{z}))$, is a function not only of the motion command but also the sensing. The simplifying assumption of Gaussian models is in general untenable for localization; however, we shall see that this assumption is sufficient for the purpose of motion planning.

One final modification must be made to the state transition rule. In a perfect world, it would be possible to predict exactly what observation would be made. However, it is exactly the stochastic and noisy nature of real sensors that generates planning difficulty, yet the update rule (6) assumes that it is possible to predict measurement $\mathbf{z}$ at pose $\mathbf{x}$. Deterministic prediction is not possible; however, it is possible to compute probabilities for sensor measurements, and thus generate an *expected value* for the entropy based on the probability distribution of observations $\mathbf{Z}$, which leads to the final state transition rule:

$$p(\mathbf{s}|u) = \langle p(\mathbf{x}|u), E_{\mathbf{Z}}[H(p(\mathbf{x}|u, \mathbf{z}))] \rangle \tag{7}$$

where $E_{\mathbf{Z}}[H(p(\mathbf{x}|u, \mathbf{z}))]$ represents the expected value of the entropy of the pose distribution over the space of possible sensor measurements.

With the transition rule in equation (7), we can now compute the transition probabilities for any particular state using a model of the robot's motion, a model of the robot's sensor and a map of the environment. The probability $p(\mathbf{x}|u)$ is given by a model of the robot's motion, and can be easily precomputed for each action $u$. The expectation term $E_{\mathbf{Z}}[H]$

can also be precomputed for each possible state s. The precomputation of these transition probabilities is very time-intensive, because it requires simulating sensing at each state in the environment, and then computing the posterior distribution. However, as the precomputation is a one-time operation for the environment and robot, planning itself can be an online operation and is (in the limit) unaffected by the speed of computing the transition probabilities.

### 3.1  Computing Trajectories

With the state update rule given in equation (7), we can now compute the optimal trajectory to a particular goal. We would in fact like to compute not just the optimal trajectory from the current robot position, but the optimal action from any position in the world. If the robot should deviate from the expected trajectory for any reason (such as error in the motion, or due to low-level control constraints), interests of efficiency suggest precomputing actions for continuing to the goal, rather than continually replanning as these contingencies arise. Note that the motion planning problem as we have now phrased it can be viewed as the problem of computing the optimal policy for a given problem. The Markovian, stochastic nature of the transitions, coupled with the need to compute the optimal policy for all states, suggests a value iteration approach.

Value iteration attempts to find the policy that maximizes the long-term reward [1, 4]. The problem becomes one of finding the value function, $J(s)$ which assigns a value to each state. The optimal action at each state can then be easily computed by determining the expected value of each action at each state, from the neighboring values. We use a modified form of Bellman's equations to give the value of state $J(s)$ and policy as

$$J(s_i) = \max_u[R(s_i) + C(s, u) + \sum_{j=1}^{N} p(s_j|s_i, u) \cdot J(s_j)] \tag{8}$$

$$\pi(s_i) = \operatorname{argmax}_u[R(s_i) + C(s, u) + \sum_{j=1}^{N} p(s_j|s_i, u) \cdot J(s_j)] \tag{9}$$

By iterating equation (8), the value function iteratively settles to a converged value over all states. Iteration stops when no state value changes above some threshold value.

In the above equations, $R(s_i)$ is the immediate reward at state $si$, $p(s_j|s_i, u)$ is the transition probability from state $s_i$ to state $s_j$, and $C(s, u)$ is the cost of taking action $u$ at state s. Note that the form of the equations is undiscounted in the traditional sense, however, the additive cost term plays a similar role in that the system is penalized for policies that take longer trajectories. The cost in general is simply the distance of one step in the given direction $u$, although the cost of travel close to obstacles is higher, in order to create a safety margin around obstacles. The cost of an action that would cause a collision is infinite, preventing such actions from being used.

The immediate reward is localized only at the goal pose. However, the goal pose has a range of possible values for the uncertainty, creating a set of goal states, $\mathcal{G}$. In order to reward policies that arrive at a goal state with a lower uncertainty, the reward is scaled linearly with goal state uncertainty.

$$R(\mathbf{x}_i) = \begin{cases} \tau - H(s) & s \in \mathcal{G} \\ 0 & \text{otherwise} \end{cases} \tag{10}$$

By implementing the value iteration given in the equations (8) and (9) in a dynamic program, we can compute the value function in $\mathcal{O}(nk_{crit})$ where $n$ is the number of states in the environment (*number of positions* × *number of entropy levels*) and $k_{crit}$ is the number of iterations to convergence. With the value function computed, we can generate the optimal action for any state in $\mathcal{O}(a)$ time, where $a$ is the number of actions out of each state.

## 4  Experimental Results

Figure 2 shows the mobile robot, Minerva, used for this research. Minerva is a RWI B-18, and senses using a 360° field of view laser range finder at 1° increments.

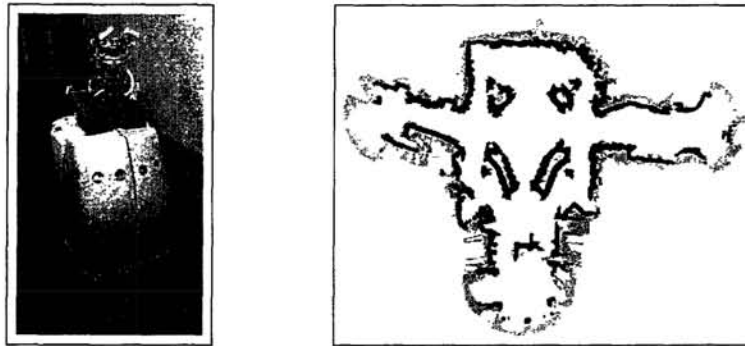

**Figure 2**: Minerva, the B-18 mobile robot used for this research, and an example environment map, the Smithsonian National Museum of American History. The black areas are the walls and obstacles. Note the large sparse areas in the center of the environment.

Also shown in figure 2 is an example environment,the Smithsonian National Museum of American History. Minerva was used to generate this map, and operated as a tour-guide in the museum for two weeks in the summer of 1998. This museum has many of the features that make localization difficult – large open spaces, and many dynamic obstacles (people) that can mislead the sensors.

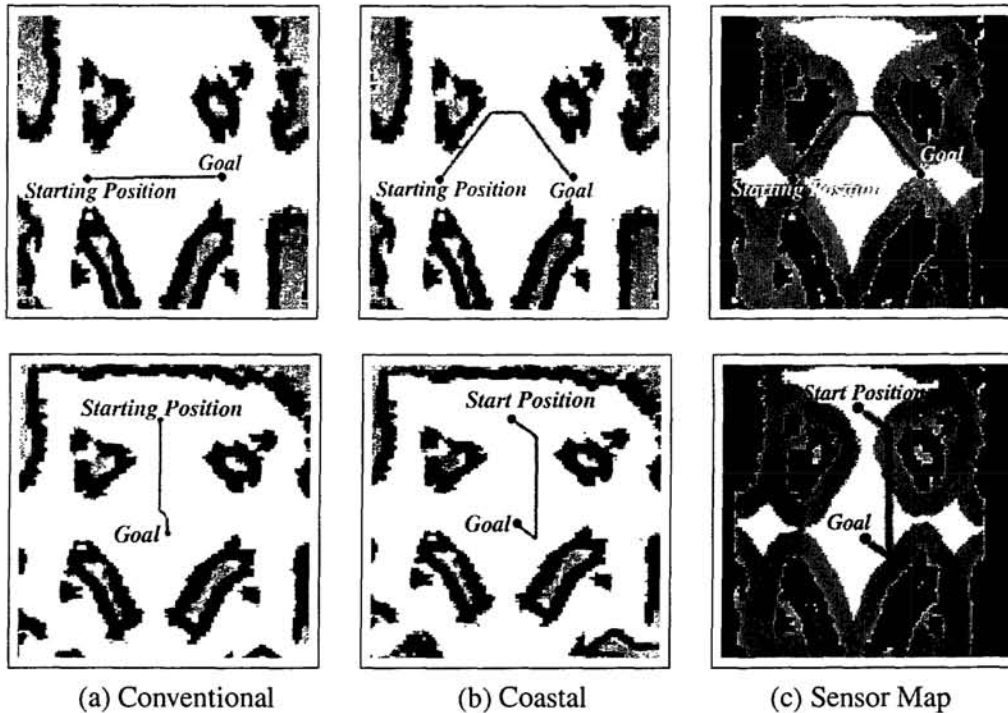

(a) Conventional          (b) Coastal          (c) Sensor Map

**Figure 3**: Two examples in the museum environment. The left trajectory is given by a conventional, shortest-path planner. The middle trajectory is given by the coastal navigation planner. The black areas correspond to obstacles, the dark grey areas correspond to regions where sensor information is available, the light grey areas to regions where no sensor information is available.

Figure 3 shows the effect of different planners in the sample environment. Panel (a) shows the trajectory of a conventional, shortest distance planner. Note that the robot moves di-

rectly towards the goal. Panel (b) shows the trajectory given by the coastal planner. In both examples, the robot moves towards an obstacle, and relocalizes once it is in sensor range of the obstacle, before moving towards the goal. These periodic relocalizations are essential for the robot to arrive at the goal with minimum positional uncertainty, and maximum reliability. Panel (c) shows the sensor map of the environment. The black areas show obstacles and walls, and the light grey areas are where no information is available to the sensors, because all environmental features are outside the range of the sensors. The dark grey areas indicate areas where the information gain from the sensors is *not* zero; the darker grey the area, the better the information gain from the sensors.

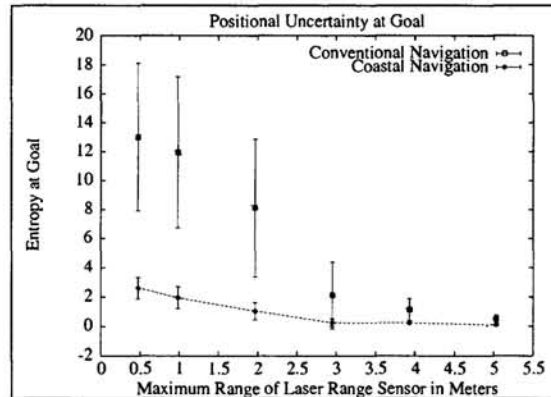

**Figure 4**: The performance of the coastal navigation algorithm compared to the coastal motion planner. The graph depicts the entropy of the position probability distribution against the range of the laser sensor. Note that the coastal navigation dramatically improves the certainty of the goal position with shorter range laser sensing.

Figure 4 is a comparison of the average positional certainty (computed as entropy of the positional probability) of the robot at its goal position, compared to the range of the laser range sensor. As the range of the laser range gets shorter, the robot can see fewer and fewer environmental features – this is essentially a way of reducing the ability of the robot to localize itself. The upper line is the performance of a conventional shortest-distance path planner, and the lower line is the coastal planner. The coastal planner has a lower uncertainty for all ranges of the laser sensor, and is substantially lower at shorter ranges, confirming that the coastal navigation has the most effect when the localization is worst.

## 5   Conclusion

In this paper, we have described a particular problem of motion planning – how to guarantee that a mobile robot can reach its goal with maximum reliability. Conventional motion planners do not typically plan according to the ability of the localization unit in different areas of the environment, and thus make no claims about the robustness of the generated trajectory. In contrast, POMDPs provide the correct solution to the problem of robust trajectories, however, computing the solution to a POMDP is intractable for the size of the state space for typical mobile robot environments.

We propose a motion planner with an augmented state space that represents positional uncertainty explicitly as an extra dimension. The motion planner then plans through pose-uncertainty space, to arrive at the goal pose with the lowest possible uncertainty. This can be seen to be an approximation to a POMDP where the multi-dimensional belief space is represented as a subset of statistics, in this case the entropy of the belief space.

We have shown some experimental comparisons with a conventional motion planner. Not only did the coastal navigation generated trajectories that provided substantial improvement of the positional certainty at the goal compared to the conventional planner, but the improvement became more pronounced as the localization was degraded.

The model presented here, however, is not complete. The entire methodology hinges upon the assumption that the robot's probability distribution can be adequately represented by the entropy of the distribution. This assumption is valid if the distribution is restricted to a uni-modal Gaussian, however, most Markov localization methods that are based on this assumption fail, because multi-modal, non-Gaussian positional distributions are quite common for moving robots. Nonetheless, it may be that multiple uncertainty statistics along multiple dimensions (e.g., $x$ and $y$) may do a better job of capturing the uncertainty sufficiently. It is an question for future work as to how many statistics can capture the uncertainty of a mobile robot, and under what environmental conditions.

## Acknowledgments

The authors gratefully acknowledge the advice and collaboration of Tom Mitchell throughout the development of this work. Wolfram Burgard and Dieter Fox played an instrumental role in the development of earlier versions of this work, and their involvement and discussion of this new model is much appreciated. This work was partially funded by the Fonds pour la Formation de Chercheurs et l'Aide à la Recherche (FCAR).

# References

[1] R. Bellman. *Dynamic Programming*. Princeton University Press, NJ, 1957.

[2] W. Burgard, D. Fox, D. Hennig, and T. Schmidt. Estimating the absolute position of a mobile robot using position probability grids. In *AAAI*, 1996.

[3] D. Fox, W. Burgard, and S. Thrun. Active Markov localization for mobile robots. *Robotics and Autonomous Systems*, 25(3-4), 1998.

[4] R. A. Howard. *Dynamic Programming and Markov Processes*. MIT, 1960.

[5] L. Kaelbling, A. R. Cassandra, and J. A. Kurien. Acting under uncertainty: Discrete Bayesian models for mobile-robot navigation. In *IROS*, 1996.

[6] S. Koenig and R. Simmons. The effect of representation and knowledge on goal-directed exploration with reinforcement learning algorithms. *Machine Learning Journal*, 22:227–250, 1996.

[7] J.-C. Latombe. *Robot Motion Planning*. Kluwer Academic Publishers, 1991.

[8] S. Mahadevan and N. Khaleeli. Robust mobile robot navigation using partially-observable semi-Markov decision processes. 1999.

[9] H. P. Moravec and A. Elfes. High resolution maps from wide angle sonar. In *ICRA*, 1985.

[10] R. Sim and G. Dudek. Mobile robot localization from learned landmarks. In *IROS*, 1998.

[11] H. Takeda, C. Facchinetti, and J.-C. Latombe. Planning the motions of mobile robot in a sensory uncertainty field. *IEEE Trans. on Pattern Analysis and Machine Intelligence*, 16(10), 1994.

[12] S. Thrun, D. Fox, and W. Burgard. A probabilistic approach to concurrent mapping and localization for mobile robots. *Machine Learning*, 431, 1998.
